# Computer Recognition of Wave Location in Graphical Data by a Neural Network

**Donald T. Freeman**
School of Medicine
University of Pittsburgh
Pittsburgh, PA 15261

## Abstract

Five experiments were performed using several neural network architectures to identify the location of a wave in the time ordered graphical results from a medical test. Baseline results from the first experiment found correct identification of the target wave in 85% of cases (n=20). Other experiments investigated the effect of different architectures and preprocessing the raw data on the results. The methods used seem most appropriate for time oriented graphical data which has a clear starting point such as electrophoresis or spectrometry rather than continuous tests such as ECGs and EEGs.

## 1 INTRODUCTION

Complex wave form recognition is generally considered to be a difficult task for machines. Analytical approaches to this problem have been described and they work with reasonable accuracy (Gabriel et al. 1980, Valdes-Sosa et al. 1987) The use of these techniques, however, requires substantial mathematical training and the process is often time consuming and labor intensive (Boston 1987). Mathematical modeling also requires substantial knowledge of the particular details of the wave forms in order to determine how to apply the models and to determine detection criteria. Rule-based expert systems have also been used for the recognition of wave forms (Boston 1989). They require that a knowledge engineer work closely with a domain expert to extract the rules that the expert uses to perform the recognition. If the rules are *ad hoc* or if it is difficult for experts to articulate the rules they use, then rule-based expert systems are cumbersome to implement.

This paper describes the use of neural networks to recognize the location of peak V from the wave-form recording of brain stem auditory evoked potential tests. General discussions of connectionist networks can be found in (Rumelhart and McClelland 1986). The main features of neural networks that are relevant for our purposes revolve around their ease of use as compared to other modeling techniques. Neural networks provide several advantages over modeling with differential equations or rule-based systems. First, there is no knowledge engineering phase. The network is trained automatically using a series of examples along with the "right answer" to each example. Second, the resulting network typically has significant predictive power when novel examples are presented. So, neural network technology allows expert performance to be mimicked without requiring that expert knowledge be codified in a traditional fashion. In addition, neural networks, when used to perform signal analysis, require vastly less restrictive

assumptions about the structure of the input signal than analytical techniques (Gorman and Sejnowski 1988). Still, neural nets have not yet been widely applied to problems of this sort (DeRoach 1989). Nevertheless, it seems that interest is growing in using computers, especially neural networks, to solve advanced problems in medical decision making (Stubbs 1988).

## 1.1 BRAIN STEM AUDITORY EVOKED POTENTIAL (BAEP)

Sensory evoked potentials are electric signals from the brain that occur in response to transient auditory, somatosensory, or visual stimuli such as a click, pinprick, or flash of light. The signals, recorded from electrodes placed on a subject's scalp, are a measure of the electrical activity in the subject's brain both from response to the stimulus and from the spontaneous electroencephalographic (EEG) activity of the brain. One way of discerning the response to the stimulus from the background EEG noise is to average the individual responses from many identical stimuli. When "cortical noise" has been removed in this way, evoked potentials can be an important noninvasive measure of central nervous system function. They are used in studies of physiology and psychology, for the diagnosis of neurologic disorders (Greenberg et al. 1981). Recently attention has focused on continuous automated monitoring of the BAEP intraoperatively as well as post-operatively for evaluation of central nervous system function (Moulton et al. 1991). Brain stem auditory evoked potentials (BAEP) are generated in the auditory pathways of the brain stem. They can be used to asses hearing and brain stem function even in unresponsive or uncooperative patients.

The BAEP test involves placing headphones on the patient, flooding one ear with white noise, and delivering clicks into the other ear. Electrodes on the scalp both on the same side (ipsilateral) and opposite side (contralateral) of the clicks record the electric potentials of brain activity for 10 msec. following each click. In the protocol used at the University of Pittsburgh Presbyterian University Hospital (PUH), a series of 2000 clicks is delivered and the results from each click - a graph of electrode activity over the 10 msec. - are averaged into a single graph. Results from the stimulation of one ear with the clicks is referred to as "one ear of data".

A graph of the wave form which results from the averaging of many stimuli appears as a series of peaks following the stimulus (Figure 1). The resulting graph typically has 7 important peaks but often includes other peaks resulting from the noise which remains after averaging. Each important peak represents the firing of a group of neurons in the auditory neural pathway[1]. The time of arrival of the peaks (the peak latencies) and the amplitudes of the peaks are used to characterize the response. The latencies of peaks I, III, and V are typically used to determine if there is evidence of slowed central nervous system conduction which is of value in the diagnosis of multiple sclerosis and other disease states[2]. Conduction delay may be seen in the left, right, or both BAEP pathways. It is of interest that the time of arrival of a wave on the ipsilateral and contralateral sides may be slightly different. This effect becomes more exagerated the more distant the correlated peaks are from the origin (Durrant, Boston, and Martin 1990).

Typically there are several issues in the interpretation of the graphs. First, it must be clear that some neural response to the auditory stimulus is represented in the wave form. If a response is present, the peaks which correspond to normal and abnormal responses must be distinguished from noise which remains in the signal even after averaging. Wave IV and wave V occasionally fuse, forming a wave IV/V complex, confounding this

process. In these cases we say that wave V is absent. Finally, the latencies and possibly the amplitudes of the identified peaks are be measured and a diagnostic explanation for them is developed.

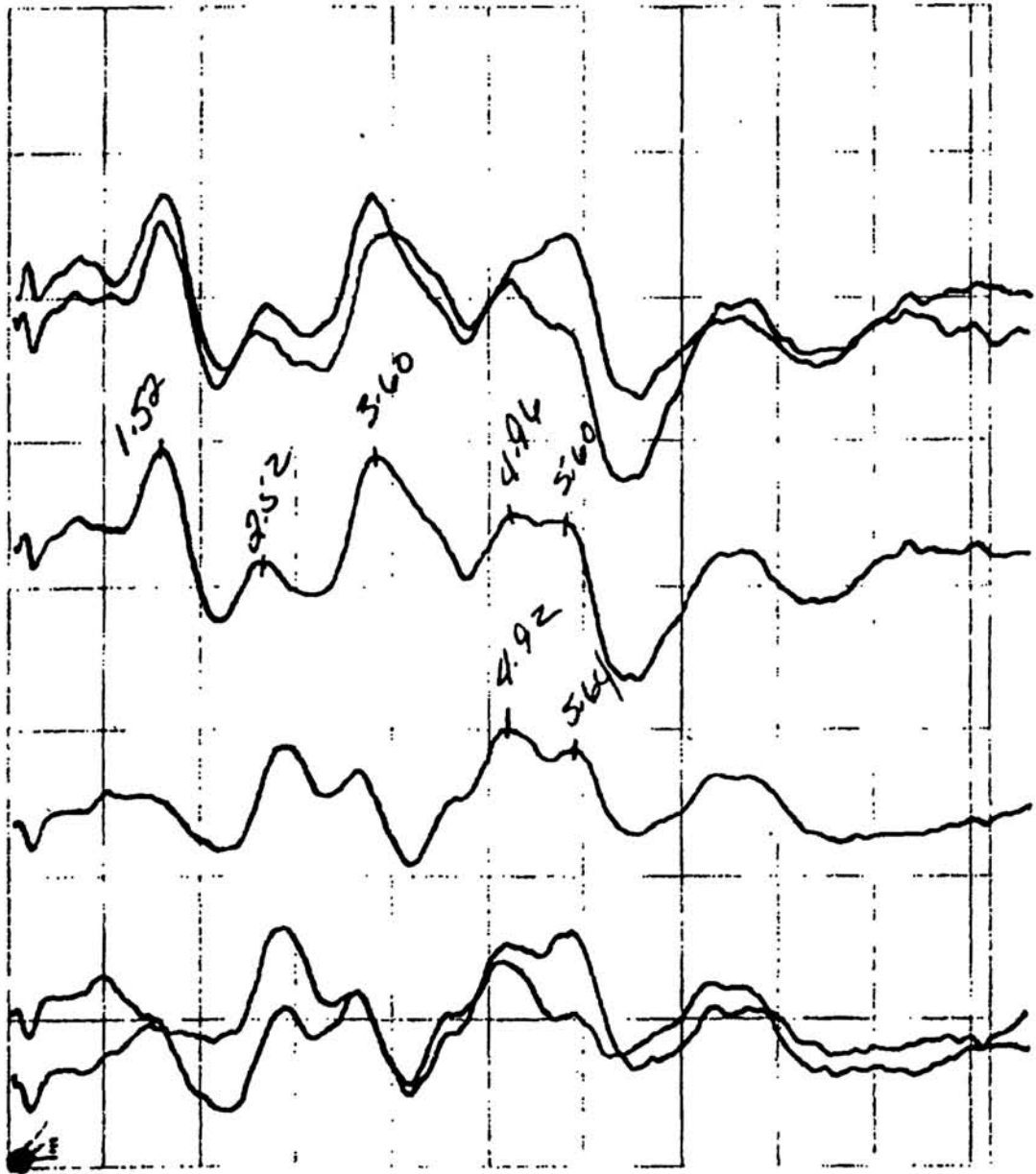

Figure 1. BAEP chart with the time of arrival for waves I to V identified.

## 2   METHODS AND PROCEDURES

### 2.1   DATA

Plots of BAEP tests were obtained from the evoked potential files from the last 4 years at PUH. A preliminary group of *training cases* consisting of 13 patients or 26 ears was selected by traversing the files alphabetically from the beginning of the alphabet. This

group was subsequently extended to 25 patients or 50 ears, 39 normals and 11 abnormals. Most BAEP tests show no abnormalities; only 1 of the first 40 ears was abnormal. In order to create a training set with an adequate number of abnormal cases we included only patients with abnormal ears after these first 40 had been selected. Ten abnormal ears were obtained from a search of 60 patient files. *Test cases* were selected from files starting at the end of the alphabet, moving toward the beginning, the opposite of the process used for the training cases. Unlike the training set - where some cases were selected over others - all cases were included in the test set without bias. No cases were common to both sets. A total of 10 patients or 20 ears were selected. Table 1 summarizes the input data.

For one of the experiments, another data set was made using the ipsilateral data for 80 inputs and the derivative of the curve for the other 80 inputs. The derivative was computed by subtracting the amplitude of the point's successor from the amplitude of the point and dividing by 0.1.

The ipsilateral and contralateral wave recordings were transformed to machine readable format by manual tracing with a BitPad Plus® digitizer. A formal protocol was followed to ensure that a high fidelity transcription had been effected. The approximately 400 points which resulted from the digitization of each ear were graphed and compared to the original tracings. If the tracings did not match, then the transcription was performed again. In addition, the originally recorded latency values for peak V were corrected for any distortion in the digitizing process. The distortion was judged by a neurologist to be minimal.

Table 1: Composition of Input Data

| Cases | Normal Ears | Abnormal Ears | | | Total Ears |
|---|---|---|---|---|---|
| | | Prolonged V | Absent V | Total | |
| Training | 39 | 8 | 3 | 11 | 50 |
| Testing | 18 | 0 | 2 | 2 | 20 |

A program was written to process the digital wave forms, creating an output file readable by the neural network simulator. The program discarded the first and last 1 msec. of the recordings. The remaining points were sampled at 0.1 msec. intervals using linear interpolation to estimate an amplitude if a point had not been recorded within 0.01 msec. of the desired time. These points were then normalized to the range <-1,1>. The resulting 80 points for the ipsilateral wave and 80 points for the contralateral wave (a total of 160 points) were used as the initial activations for the input layer of processing elements.

## 2.2 ARCHITECTURES

Each of the four network architectures had 160 input nodes. Each node represented the amplitude of the wave at each sample time (1.0 to 8.9 ms, every 0.1 ms). Each architecture also had 80 output nodes with a similar temporal interpretation (Figure 2). Architecture 1 (A1) had 30 hidden units connected only to the ipsilateral input units, 5 hidden units connected only to the contralateral input units and 5 hidden units connected to all the input units. The hidden units for all architectures were fully connected to the output units. Architecture 2 (A2) reversed these proportions. Architecture 3 (A3) was fully connected to the inputs. Architecture 4 (A4) preserved the proportions of A1 but had 16 ipsilateral hidden units, 3 contralateral, and 3 connected to both. All architectures used the sigmoid transfer function at both the hidden and output layers and all units were attached to a bias unit.

The distribution of the hidden units was chosen with the knowledge that human experts usually use information from the ipsilateral side but refer to the contralateral side only

when features in the ipsilateral side are too obscure to resolve. The selection of the number of hidden units in neural network models remains an art. In order to determine whether the size of the hidden unit layer could be changed, we repeated the experiments using Architecture 2 where the number of hidden units was reduced to 16, with 10 connected to the ipsilateral inputs, 3 to the contralateral inputs, and 3 connected to all the inputs.

## 2.3   TRAINING

For training, target values for the output layer were all 0.0 except for the output nodes representing the time of arrival for wave V (reported on the BAEP chart) and one node on each side of it. The peak node target was 0.95 and the two adjacent nodes had targets of 0.90. For cases in which wave V was absent, the target for all the output nodes was 0.0.

A neural network simulator (NeuralWorks Professional II® version 3.5) was used to construct the networks and run the simulations. The back-propagation learning algorithm was used to train the networks. The random number generator was initialized with random number seeds taken from a random number table. Then network weights were initialized to random values between -0.2 and 0.2 and the training begun. Since our random number generator is deterministic – given the random number seed – these trials are replicable.

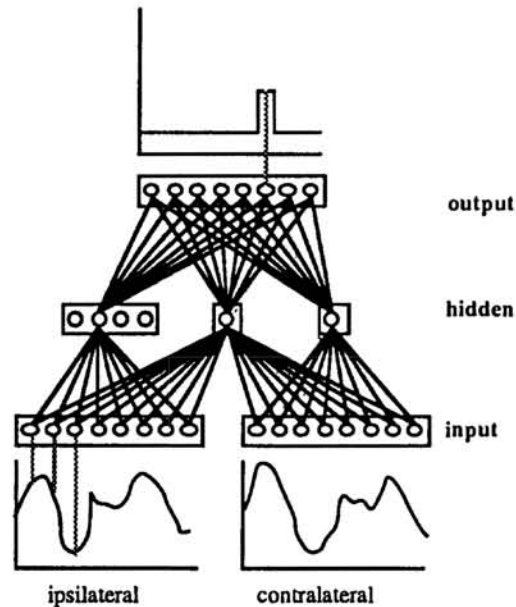

Figure 2. Diagram of Architecture 1 with representation of input and output data shown.

Each of the 50 ears of data in the training set was presented using a randomize, shuffle, and deal technique. Network weights were saved at various stages of learning, usually after every 1000 presentations (20 epochs) until the cumulative RMS error for an epoch fell below 0.01. The contribution of each training example to the total error was examined to determine whether a few examples were the source of most of the error. If so, training was continued until these examples had been learned to an error level comparable to the rest of the cases. After training, the 20 ears in the test set were presented to each of the saved networks and the output nodes of the net were examined for each test case.

## 2.4   ANALYSIS OF RESULTS

A threshold method was used to analyze the data. For each of the test cases the actual location of the maximum valued output unit was compared to the expected location of the maximum valued output unit. For a network result to be classified as a correct identification in the wave V present (true positive), we require that the maximum valued output unit have an activation which is over an activity-threshold (0.50) and that the unit be within a distance-threshold (0.2 msec.) of the expected location of wave V. For a true negative identification of wave V – a correct identification of wave V being absent – we require that all the output activities be below the activity threshold and that the case have no wave V to find. The network makes a false positive prediction of the location of wave V if some activity is above the activity threshold for a case which has no wave V. Finally, there are two ways for the network to make a false negative identification of wave V. In both instances, wave V must be present in the case. In one instance, some output node has activity above the activity threshold, but it is outside of the distance threshold. This corresponds to the identification of a wave V but in the wrong place. In the other instance, no node attains activity over the activity threshold, corresponding to a failure to find a wave V when there exists a wave V in the case to find.

## 2.5   EXPERIMENTS

Five experiments were performed. The first four used different architectures on the same data set and the last used architecture A1 on the derivatives data set. Each of the network architectures was trained from different random starting positions. For each trial, a network was randomized and trained as described above. The networks were sampled as learning progressed.

Experiment 1 determined how well architecture A1 could identify wave V and provided baseline results for the remaining experiments. Experiments 2 and 3 tested whether our use of more hidden units attached to ipsilateral data made sense by reversing the proportion of hidden units alloted to ipsilateral data processing (experiment 2) and by tring a fully connected network (experiment 3). Experiment 4 determined whether fewer hidden units could be used. Experiment 5 investigated whether preprocessing of the input data to make derivative information available would facilitate network identification of peak location.

## 3   RESULTS

Results from the best network found for each of five experiments are shown in Table 2.

Table 2: Results from presentation of 20 test cases to various network architectures.

| Experiment | Network | TP | TN | Total | FP | FN | Total |
|------------|---------|----|----|-------|----|----|-------|
| 1 | A1 | 16 | 1 | 17 | 1 | 2 | 3 |
| 2 | A2 | 16 | 0 | 16 | 2 | 2 | 4 |
| 3 | A3 | 16 | 0 | 16 | 2 | 2 | 4 |
| 4 | A4 | 15 | 0 | 15 | 3 | 2 | 5 |
| 5 | A1 | 15 | 1 | 16 | 1 | 3 | 4 |

## 4   DISCUSSION

In Experiment 1, the three cases which were incorrectly identified were examined closely. It is not evident from inspection why the net failed to identify the peaks or identified

peaks where there were none to identify. Where peaks are present, they are not unusually located or surrounded by noise. The appearance of their shape seems similar to the cases which were identified correctly. We believe that more training examples which are "similar" to these 3 test cases, as well as examples with greater variety, will improve recognition of these cases. This improvement comes not from better generalization but rather from a reduced requirement for generalization. If the net is trained with cases which are increasingly similar to the cases which will be used to test it, then recognition of the test cases becomes easier at any given level of generalization.

The distribution of hidden units in A1 was chosen with the knowledge that human experts use information primarily from the ipsilateral side, referring to the contralateral side only when ipsilateral features are too obscure to resolve. Experiments 2 and 3 investigate whether this reliance on ipsilateral data suggests that there should be more hidden units for the ipsilateral side or for the contralateral side. The identical results from these experiments are similar to those of Experiment 1. One interpretation is that it is possible to make diagnoses of BAEPs using very few features from the ipsilateral side. Another interpretation is that it is possible to use the contralateral data as the chief information source, contrary to our expert's belief.

Experiment 4 investigates whether fewer features are needed by restricting the hidden layer to 20 hidden units. The slight degradation of performance indicates that it is possible to make BAEP diagnoses with fewer ipsilateral features. Experiment 5 utilized the ipsilateral waveform and its derivative to determine whether this pre-processing would improve the results. Surprisingly, the results did not improve, but it is possible that a better estimator of the derivative will prove this method useful.

Finally, when the weights from all the networks above were examined, we found that amplitudes from only the area where wave V falls were used. This suggests that it is not necessary to know the location of wave III before determining the location of wave V, in sharp contrast to expert's intuition. We believe the networks form a "local expert" for the identification of wave V which does not need to interact with data from other parts of the graph, and that other such local experts will be formed as we expand the project's scope.

## 5   CONCLUSIONS

Automated wave form recognition is considered to be a difficult task for machines and an especially difficult task for neural networks. Our results offer some encouragement that in some domains neural networks may be applied to perform wave form recognition and that the technique will be extensible as problem complexity increases.

Still, the accuracy of the networks we have discussed is not high enough for clinical use. Several extensions have been attempted and others considered including 1) increasing the sampling rate to decrease the granularity of the input data, 2) increasing the training set size, 3) using a different representation of the output for wave V absent cases, 4) using a different representation of the input, such as the derivative of the amplitudes, and 5) architectures which allow hybrids of these ideas.

Finally, since many other tests in medicine as well as other fields require the interpretation of graphical data, it is tempting to consider extending this method to other domains. Many other tests in medicine as well as other fields require the interpretation of graphical data.One distinguishing feature of the BAEP is that there is no difficulty with the time registration of the data; we always know where to start looking for the wave. This is in contrast to an EKG, for example, which may require substantial effort just to identify the beginning of a QRS complex. Our results indicate that the interpretation of graphs where the time registration of data is not an issue is possible using neural networks. Medical tests for which this technique would be appropriate include: other evoked potentials, spectrometry, and gel electrophoresis.

## Acknowledgements

The author wishes to thank Dr. Scott Shoemaker of the Department of Neurology for his expertise, encouragement, constructive criticism, patience, and collaboration throughout the progress of this work. This research has been supported by NLM Training grant T15 LM-07059.

## Footnotes

[1]Putative generators are: I–Acoustic nerve; II–Cochlear nucleus; III–Superior olivary nucleus; IV–Lateral lemniscus; V-Inferior colliculus; VI–Medial geniculate nucleus; VII–Auditory radiations.

[2]Other disorders include brain edema, acoustic neuroma, gliomas, and central pontine myelinolysis.

## References

Boston, J.R. 1987. Detection criteria for sensory evoked potentials. Proceedings of 9th Ann. IEEE/EMBS Conf., Boston, MA.

Boston, J.R. 1989. Automated interpretation of brainstem auditory evoked potentials: a prototype system. IEEE Trans. Biomed. Eng. 36 (5) : 528-532.

DeRoach, J.N. 1989. Neural networks - an artificial intelligence approach to the analysis of clinical data. Austral. Phys. & Eng. Sci. in Med. 12 (2) : 100-106.

Durrant, J.D., J.R. Boston, and W.H. Martin. 1990. Correlation study of two-channel recordings of the brain stem auditory evoked potential. Ear and Hearing 11 (3) : 215-221.

Gabriel, S., J.D. Durrant, A.E. Dickter, and J.E. Kephart. 1980. Computer identification of waves in the auditory brain stem evoked potentials. EEG and Clin. Neurophys. 49 : 421-423.

Gorman, R. Paul, and Terrence J. Sejnowski. 1988. Analysis of hidden units in a layered network trained to classify sonar targets. Neural Networks 1 : 75-89.

Greenberg, R.P., P.G. Newlon, M.S. Hyatt, R.K. Narayan, and D.P. Becker. 1981. Prognostic implications of early multimodality evoked potentials in severely head-injured patients. J. Neurosurg 5 : 227-236.

Moulton, Richard, Peter Kresta, Mario Ramirez, and William Tucker. 1991. Continuous automated monitoring of somatosensory evoked potentials in posttraumatic coma. Journal of Trauma 31 (5) : 676-685.

Rumelhart, David E., and James L. McClelland. 1986. Parallel distributed processing. Cambridge, Mass: MIT Press.

Stubbs, D F. 1988. Neurocomputers. MD Comput 5 (3) : 14-24.

Valdes-Sosa, M.J., M.A. Bobes, M.C. Perez-abalo, M. Perra, J.A. Carballo, and P. Valdes-Sosa. 1987. Comparison of auditory evoked potential detection methods using signal detection theory. Audiol 26 : 166-178.